# An Infinity-sample Theory for Multi-category Large Margin Classification

**Tong Zhang**
IBM T.J. Watson Research Center
Yorktown Heights, NY 10598
tzhang@watson.ibm.com

## Abstract

The purpose of this paper is to investigate infinity-sample properties of risk minimization based multi-category classification methods. These methods can be considered as natural extensions to binary large margin classification. We establish conditions that guarantee the infinity-sample consistency of classifiers obtained in the risk minimization framework. Examples are provided for two specific forms of the general formulation, which extend a number of known methods. Using these examples, we show that some risk minimization formulations can also be used to obtain conditional probability estimates for the underlying problem. Such conditional probability information will be useful for statistical inferencing tasks beyond classification.

## 1 Motivation

Consider a binary classification problem where we want to predict label $y \in \{\pm 1\}$ based on observation $x$. One of the most significant achievements for binary classification in machine learning is the invention of large margin methods, which include support vector machines and boosting algorithms. Based on a set of observations $(X_1, Y_1), \ldots, (X_n, Y_n)$, a large margin classification algorithm produces a decision function $\hat{f}_n$ by empirically minimizing a loss function that is often a convex upper bound of the binary classification error function. Given $\hat{f}_n$, the binary decision rule is to predict $y = 1$ if $\hat{f}_n(x) \geq 0$, and to predict $y = -1$ otherwise (the decision rule at $\hat{f}_n(x) = 0$ is not important). In the literature, the following form of large margin binary classification is often encountered: we minimize the empirical risk associated with a convex function $\phi$ in a pre-chosen function class $C_n$:

$$\hat{f}_n = \arg \min_{f \in C_n} \frac{1}{n} \sum_{i=1}^{n} \phi(f(X_i)Y_i). \tag{1}$$

Originally such a scheme was regarded as a compromise to avoid computational difficulties associated with direct classification error minimization, which often leads to an NP-hard problem. The current view in the statistical literature interprets such methods as algorithms to obtain conditional probability estimates. For example, see [3, 6, 9, 11] for some related studies. This point of view allows people to show the consistency of various large margin

methods: that is, in the large sample limit, the obtained classifiers achieve the optimal Bayes error rate. For example, see [1, 4, 7, 8, 10, 11]. The consistency of a learning method is certainly a very desirable property, and one may argue that a good classification method should be consistent in the large sample limit.

Although statistical properties of binary classification algorithms based on the risk minimization formulation (1) are quite well-understood due to many recent works such as those mentioned above, there are much fewer studies on risk minimization based multi-category problems which generalizes the binary large margin method (1). The complexity of possible generalizations may be one reason. Another reason may be that one can always estimate the conditional probability for a multi-category problem using the binary classification formulation (1) for each category, and then pick the category with the highest estimated conditional probability (or score).[1] However, it is still useful to understand whether there are more natural alternatives, and what kind of risk minimization formulation which generalizes (1) can be used to yield consistent classifiers in the large sample limit. An important step toward this direction has recently been taken in [5], where the authors proposed a multi-category extension of the support vector machine that is Bayes consistent (note that there were a number of earlier proposals that were not consistent). The purpose of this paper is to generalize their investigation so as to include a much wider class of risk minimization formulations that can lead to consistent classifiers in the infinity-sample limit. We shall see that there is a rich structure in risk minimization based multi-category classification formulations. Multi-category large margin methods have started to draw more attention recently. For example, in [2], learning bounds for some multi-category convex risk minimization methods were obtained, although the authors did not study possible choices of Bayes consistent formulations.

## 2 Multi-category classification

We consider the following $K$-class classification problem: we would like to predict the label $y \in \{1, \ldots, K\}$ of an input vector $x$. In this paper, we only consider the simplest scenario with $0 - 1$ classification loss: we have a loss of 0 for correct prediction, and loss of 1 for incorrect prediction.

In binary classification, the class label can be determined using the sign of a decision function. This can be generalized to $K$ class classification problem as follows: we consider $K$ decision functions $f_c(x)$ where $c = 1, \ldots, K$ and we predict the label $y$ of $x$ as:

$$T(f(x)) = \arg \max_{c \in \{1, \ldots, K\}} f_c(x), \tag{2}$$

where we denote by $f(x)$ the vector function $f(x) = [f_1(x), \ldots, f_K(x)]$.

Note that if two or more components of $f$ achieve the same maximum value, then we may choose any of them as $T(f)$. In this framework, $f_c(x)$ is often regarded as a scoring function for category $c$ that is correlated with how likely $x$ belongs to category $c$ (compared with the remaining $k - 1$ categories). The classification error is given by:

$$\ell(f) = 1 - E_X P(Y = T(X)|X).$$

Note that only the relative strength of $f_c$ compared with the alternatives is important. In particular, the decision rule given in (2) does not change when we add the same numerical quantity to each component of $f(x)$. This allows us to impose one constraint on the vector $f(x)$ which decreases the degree of freedom $K$ of the $K$-component vector $f(x)$ to $K - 1$.

For example, in the binary classification case, we can enforce $f_1(x) + f_2(x) = 0$, and hence $f(x)$ can be represented as $[f_1(x), -f_1(x)]$. The decision rule in (2), which compares $f_1(x) \geq f_2(x)$, is equivalent to $f_1(x) \geq 0$. This leads to the binary classification rule mentioned in the introduction.

In the multi-category case, one may also interpret the possible constraint on the vector function $f$, which reduces its degree of freedom from $K$ to $K-1$ based on the following reasoning. In many cases, we seek $f_c(x)$ as a function of $p(Y = c|x)$. Since we have a constraint $\sum_{c=1}^{K} p(Y = c|x) = 1$ (implying that the degree of freedom for $p(Y = c|x)$ is $K-1$), the degree of freedom for $f$ is also $K-1$ (instead of $K$). However, we shall point out that in the algorithms we formulate below, we may either enforce such a constraint that reduces the degree of freedom of $f$, or we do not impose any constraint, which keeps the degree of freedom of $f$ to be $K$. The advantage of the latter is that it allows the computation of each $f_c$ to be decoupled. It is thus much simpler both conceptually and numerically. Moreover, it directly handles multiple-label problems where we may assign each $x$ to multiple labels of $y \in \{1, \dots, K\}$. In this scenario, we do not have a constraint.

In this paper, we consider an empirical risk minimization method to solve a multi-category problem, which is of the following general form:

$$\hat{f}_n = \arg \min_{f \in C_n} \frac{1}{n} \sum_{i=1}^{n} \Psi_{Y_i}(f(X_i)). \qquad (3)$$

As we shall see later, this method is a natural generalization of the binary classification method (1). Note that one may consider an even more general form with $\Psi_Y(f(X))$ replaced by $\Psi_Y(f(X), X)$, which we don't study in this paper.

From the standard learning theory, one can expect that with appropriately chosen $C_n$, the solution $\hat{f}_n$ of (3) approximately minimizes the true risk $R(\hat{f})$ with respect to the unknown underlying distribution within the function class $C_n$,

$$R(f) = \mathbf{E}_{X,Y} \Psi_Y(f(X)) = \mathbf{E}_X L(P(\cdot|X), f(X)), \qquad (4)$$

where $P(\cdot|X) = [P(Y = 1|X), \dots, P(Y = K|X)]$ is the conditional probability, and

$$L(q, f) = \sum_{c=1}^{K} q_c \Psi_c(f). \qquad (5)$$

In order to understand the large sample behavior of the algorithm based on solving (3), we first need to understand the behavior of a function $f$ that approximately minimizes $R(f)$. We introduce the following definition (also referred to as classification calibrated in [1]):

**Definition 2.1** *Consider $\Psi_c(f)$ in (4). We say that the formulation is admissible (classification calibrated) on a closed set $\Omega \subseteq [-\infty, \infty]^K$ if the following conditions hold: $\forall c$, $\Psi_c(\cdot) : \Omega \to (-\infty, \infty]$ is bounded below and continuous; $\cap_c \{f : \Psi_c(f) < \infty\}$ is non-empty and dense in $\Omega$; $\forall q$, if $L(q, f^*) = \inf_f L(q, f)$, then $f_c^* = \sup_k f_k^*$ implies $q_c = \sup_k q_k$.*

Since we allow $\Psi_c(f) = \infty$, we use the convention that $q_c \Psi_c(f) = 0$ when $q_c = 0$ and $\Psi_c(f) = \infty$. The following result relates the approximate minimization of the $\Psi$ risk to the approximate minimization of classification error:

**Theorem 2.1** *Let $\mathcal{B}$ be the set of all Borel measurable functions. For a closed set $\Omega \subset [-\infty, \infty]^K$, let $\mathcal{B}_\Omega = \{f \in \mathcal{B} : \forall x, f(x) \in \Omega\}$. If $\Psi_c(\cdot)$ is admissible on $\Omega$, then for a Borel measurable distribution, $R(f) \to \inf_{g \in \mathcal{B}_\Omega} R(g)$ implies $\ell(f) \to \inf_{g \in \mathcal{B}} \ell(g)$.*

*Proof Sketch.* First we show that the admissibility implies that $\forall \epsilon > 0$, $\exists \delta > 0$ such that $\forall q$ and $x$:

$$\inf_{q_c \leq \sup_k q_k - \epsilon} \{L(q,f) : f_c = \sup_k f_k\} \geq \inf_{g \in \Omega} L(q,g) + \delta. \tag{6}$$

If (6) does not hold, then $\exists \epsilon > 0$, and a sequence of $(c^m, f^m, q^m)$ with $f^m \in \Omega$ such that $f^m_{c^m} = \sup_k f^m_k$, $q^m_{c^m} \leq \sup_k q^m_k - \epsilon$, and $L(q^m, f^m) - \inf_{g \in \Omega} L(q^m, g) \to 0$. Taking a limit point of $(c^m, f^m, q^m)$, and using the continuity of $\Psi_c(\cdot)$, we obtain a contradiction (technical details handling the infinity case are skipped). Therefore (6) must be valid.

Now we consider a vector function $f(x) \in \Omega_{\mathcal{B}}$. Let $q(x) = P(\cdot|x)$. Given $X$, if $P(Y = T(f(X))|X) \geq P(Y = T(q(X))|X) + \epsilon$, then equation (6) implies that $L(q(X), f(X)) \geq \inf_{g \in \Omega} L(q(X), g) + \delta$. Therefore

$$
\begin{aligned}
\ell(f) - \inf_{g \in \mathcal{B}} \ell(g) &= E_X[P(Y = T(q(X))|X) - P(Y = T(f(X))|X)] \\
&\leq \epsilon + E_X I(P(Y = T(q(X))|X) - P(Y = T(f(X))|X) > \epsilon) \\
&\leq \epsilon + E_X \frac{L_X(q(X), f(X)) - \inf_{g \in \mathcal{B}_\Omega} L_X(q(X), g)}{\delta} \\
&= \epsilon + \frac{R(f) - \inf_{g \in \mathcal{B}_\Omega} R(g)}{\delta}.
\end{aligned}
$$

In the above derivation we use $I$ to denote the indicator function. Since $\epsilon$ and $\delta$ are arbitrary, we obtain the theorem by letting $\epsilon \to 0$. $\square$

Clearly, based on the above theorem, an admissible risk minimization formulation is suitable for multi-category classification problems. The classifier obtained from minimizing (3) can approach the Bayes error rate if we can show that with appropriately chosen function class $C_n$, approximate minimization of (3) implies approximate minimization of (4). Learning bounds of this forms have been very well-studied in statistics and machine learning. For example, for large margin binary classification, such bounds can be found in [4, 7, 8, 10, 11, 1], where they were used to prove the consistency of various large margin methods. In order to achieve consistency, it is also necessary to take a sequence of function classes $C_n$ $(C_1 \subset C_2 \subset \cdots)$ such that $\cup_n C_n$ is dense in the set of Borel measurable functions. The set $C_n$ has the effect of regularization, which ensures that $R(\hat{f}_n) \approx \inf_{f \in C_n} R(f)$. It follows that as $n \to \infty$, $R(\hat{f}_n) \xrightarrow{P} \inf_{f \in \mathcal{B}} R(f)$. Theorem 2.1 then implies that $\ell(\hat{f}_n) \xrightarrow{P} \inf_{f \in \mathcal{B}} \ell(f)$.

The purpose of this paper is not to study similar learning bounds that relate approximate minimization of (3) to the approximate minimization of (4). See [2] for a recent investigation. We shall focus on the choices of $\Psi$ that lead to admissible formulations. We pay special attention to the case that each $\Psi_c(f)$ is a convex function of $f$, so that the resulting formulation becomes computational more tractable. Instead of working with the general form of $\Psi_c$ in (4), we focus on two specific choices listed in the next two sections.

## 3 Unconstrained formulations

We consider unconstrained formulation with the following choice of $\Psi$:

$$\Psi_c(f) = \phi(f_c) + s\left(\sum_{k=1}^K t(f_k)\right), \tag{7}$$

where $\phi$, $s$ and $t$ are appropriately chosen functions that are continuously differentiable.

The first term, which has a relatively simple form, depends on the label $c$. The second term is independent of the label, and can be regarded as a normalization term. Note that

this function is symmetric with respect to components of $f$. This choice treats all potential classes equally. It is also possible to treat different classes differently (e.g. replacing $\phi(f_c)$ by $\phi_c(f_c)$), which can be useful if we associate different classification loss to different kinds of errors.

## 3.1 Optimality equation and probability model

Using (7), the conditional true risk (5) can be written as:

$$L(q, f) = \sum_{c=1}^{K} q_c \phi(f_c) + s\left(\sum_{c=1}^{K} t(f_c)\right).$$

In the following, we study the property of the optimal vector $f^*$ that minimizes $L(q, f)$ for a fixed $q$. Given $q$, the optimal solution $f^*$ of $L(q, f)$ satisfies the following first order condition:

$$q_c \phi'(f_c^*) + \mu_{f^*} t'(f_c^*) = 0 \qquad (c = 1, \ldots, K). \tag{8}$$

where quantity $\mu_{f^*} = s'(\sum_{k=1}^{K} t(f_k^*))$ is independent of $k$.

Clearly this equation relates $q_c$ to $f_c^*$ for each component $c$. The relationship of $q$ and $f^*$ defined by (8) can be regarded as the (infinite sample-size) probability model associated with the learning method (3) with $\Psi$ given by (7).

The following result presents a simple criterion to check admissibility. We skip the proof for simplicity. Most of our examples satisfy the condition.

**Proposition 3.1** *Consider (7). Assume $\Phi_c(f)$ is continuous on $[-\infty, \infty]^K$ and bounded below. If $s'(u) \geq 0$ and $\forall p > 0$, $p\phi'(f) + t'(f) = 0$ has a unique solution $f_p$ that is an increasing function of $p$, then the formulation is admissible.*

If $s(u) = u$, the condition $\forall p > 0$ in Proposition 3.1 can be replaced by $\forall p \in (0, 1)$.

## 3.2 Decoupled formulations

We let $s(u) = u$ in (7). The optimality condition (8) becomes

$$q_c \phi'(f_c^*) + t'(f_c^*) = 0 \qquad (c = 1, \ldots, K). \tag{9}$$

This means that we have $K$ decoupled equalities, one for each $f_c$. This is the simplest and in the author's opinion, the most interesting formulation. Since the estimation problem in (3) is also decoupled into $K$ separate equations, one for each component of $\hat{f}_n$, this class of methods are computationally relatively simple and easy to parallelize. Although this method seems to be preferable for multi-category problems, it is not the most efficient way for two-class problem (if we want to treat the two classes in a symmetric manner) since we have to solve two separate equations. We only need to deal with one equation in (1) due to the fact that an effective constraint $f_1 + f_2 = 0$ can be used to reduce the number of equations. This variable elimination has little impact if there are many categories.

In the following, we list some examples of multi-category risk minimization formulations. They all satisfy the admissibility condition in Proposition 3.1. We focus on the relationship of the optimal optimizer function $f_*(q)$ and the conditional probability $q$. For simplicity, we focus on the choice $\phi(u) = -u$.

### 3.2.1 $\phi(u) = -u$ and $t(u) = e^u$

We obtain the following probability model: $q_c = e^{f_c^*}$. This formulation is closely related to the maximum-likelihood estimate with conditional model $q_c = e^{f_c} / \sum_{k=1}^{K} e^{f_k}$ (logistic

regression). In particular, if we choose a function class such that the normalization condition $\sum_{k=1}^{K} e^{f_k} = 1$ holds, then the two formulations are identical. However, they become different when we do not impose such a normalization condition.

Another very important and closely related formulation is the choice of $\phi(u) = -\ln u$ and $t(u) = u$. This is an extension of maximum-likelihood estimate with probability model $q_c = f_c$. The resulting method is identical to maximum-likelihood if we choose our function class such that $\sum_k f_k = 1$. However, the formulation also allows us to use function classes that do not satisfy the normalization constraint $\sum_k f_k = 1$. Therefore this method is more flexible.

### 3.2.2 $\phi(u) = -u$ and $t(u) = \ln(1 + e^u)$

This version uses binary logistic regression loss, and we have the following probability model: $q_c = (1 + e^{-f_c^*})^{-1}$. Again this is an unnormalized model.

### 3.2.3 $\phi(u) = -u$ and $t(u) = \frac{1}{p}|u|^p$ $(p > 1)$

We obtain the following probability model: $q_c = \text{sign}(f_c^*)|f_c^*|^{p-1}$. This means that at the solution, $f_c^* \geq 0$. One may modify it such that we allow $f_c^* \leq 0$ to model the condition probability $q_c = 0$.

### 3.2.4 $\phi(u) = -u$ and $t(u) = \frac{1}{p}\max(u, 0)^p$ $(p > 1)$

In this probability model, we have the following relationship: $q_c = \max(f_c^*, 0)^{p-1}$. The equation implies that we allow $f_c^* \leq 0$ to model the conditional probability $q_c = 0$. Therefore, with a fixed function class, this model is more powerful than the previous one. However, at the optimal solution, $f_c^* \leq 1$. This requirement can be further alleviated with the following modification.

### 3.2.5 $\phi(u) = -u$ and $t(u) = \frac{1}{p}\min(\max(u, 0)^p, p(u-1)+1)$ $(p > 1)$

In this probability model, we have the following relationship at the exact solution: $q_c = \min(\max(f_*^c, 0), 1)^{p-1}$. Clearly this model is more powerful than the previous model since the function value $f_c^* \geq 1$ can be used to model $q_c = 1$.

## 3.3 Coupled formulations

In the coupled formulation with $s(u) \neq u$, the probability model can be normalized in a certain way. We list a few examples.

### 3.3.1 $\phi(u) = -u$, and $t(u) = e^u$, and $s(u) = \ln(u)$

This is the standard logistic regression model. The probability model is:

$$q_c(x) = \exp(f_c^*(x))(\sum_{c=1}^{K} \exp(f_c^*(x)))^{-1}.$$

The right hand side is always normalized (sum up to 1). Note that the model is not continuous at infinities, and thus not admissible in our definition. However, we may consider the region $\Omega = \{f : \sup_k f_k = 0\}$, and it is easy to check that this model is admissible in $\Omega$. Let $f_c^\Omega = f_c - \sup_k f_k \in \Omega$, then $f^\Omega$ has the same decision rule as $f$ and $R(f) = R(f^\Omega)$. Therefore Theorem 2.1 implies that $R(f) \to \inf_{g \in \mathcal{B}} R(g)$ implies $\ell(f) \to \inf_{g \in \mathcal{B}} \ell(g)$.

**3.3.2** $\phi(u) = -u$, **and** $t(u) = |u|^{p'}$, **and** $s(u) = \frac{1}{p}|u|^{p/p'}$ $(p, p' > 1)$

The probability model is:

$$q_c(x) = (\sum_{k=1}^{K} |f_k^*(x)|^{p'})^{(p-p')/p'} \operatorname{sign}(f_c^*(x))|f_c^*(x)|^{p'-1}.$$

We may replace $t(u)$ by $t(u) = \max(0, u)^p$, and the probability model becomes:

$$q_c(x) = (\sum_{k=1}^{K} \max(f_k^*(x), 0)^{p'})^{(p-p')/p'} \max(f_c^*(x), 0)^{p'-1}.$$

These formulations do not seem to have advantages over the decoupled counterparts. Note that if we let $p \to 1$, then the sum of the $\frac{p'}{p'-1}$-th power of the right hand side $\to 1$. In a way, this means that the model is normalized in the limit of $p \to 1$.

## 4 Constrained formulations

As pointed out, one may impose constraints on possible choices of $f$. We may impose such a condition when we specify the function class $C_n$. However, for clarity, we shall directly impose a condition into our formulation. If we impose a constraint into (7), then its effect is rather similar to that of the second term in (7). In this section, we consider a direct extension of binary large-margin method (1) to multi-category case. The choice given below is motivated by [5], where an extension of SVM was proposed. We use a risk formulation that is different from (7), and for simplicity, we will consider linear equality constraint only:

$$\Psi_c(f) = \sum_{k=1, k \neq c}^{K} \phi(-f_k), \qquad \text{s.t.} \quad f \in \Omega, \tag{10}$$

where we define $\Omega$ as:

$$\Omega = \{f : \sum_{k=1}^{K} f_k = 0\} \cup \{f : \sup_k f_k = \infty\}.$$

We may interpret the added constraint as a restriction on the function class $C_n$ in (3) such that every $f \in C_n$ satisfies the constraint. Note that with $K = 2$, this leads to the usually binary large margin method. Using (10), the conditional true risk (5) can be written as:

$$L(q, f) = \sum_{c=1}^{K} (1 - q_c)\phi(-f_c), \quad \text{s.t. } f \in \Omega. \tag{11}$$

The following result provides a simple way to check the admissibility of (10).

**Proposition 4.1** *If $\phi$ is a convex function which is bounded below and $\phi'(0) < 0$, then (10) is admissible on $\Omega$.*

*Proof Sketch.* The continuity condition is straight-forward to verify. We may also assume that $\phi(\cdot) \geq 0$ without loss of generality. Now let $f$ achieves the minimum of $L(q, \cdot)$. If $f_c = \infty$, then it is clear that $q_c = 1$ and thus $q_k = 0$ for $k \neq c$. This implies that for $k \neq c$, $\phi(-f_k) = \inf_f \phi(-f)$, and thus $f_k < 0$. If $f_c = \sup_k f_k < \infty$, then the constraint implies $f_c \geq 0$. It is easy to see that $\forall k$, $q_c \geq q_k$ since otherwise, we must have $\phi(-f_k) > \phi(-f_c)$, and thus $\phi'(-f_k) > 0$ and $\phi'(-f_c) < 0$, implying that with sufficient small $\delta > 0$, $\phi(-(f_k + \delta)) < \phi(-f_k)$ and $\phi(-(f_c - \delta)) < \phi(-f_c)$. A contradiction. $\square$

Using the above criterion, we can convert any admissible convex $\phi$ for the binary formulation (1) into an admissible multi-category classification formulation (10).

In [5] the special case of SVM (with loss function $\phi(u) = \max(0, 1-u)$) was studied. The authors demonstrated the admissibility by direct calculation, although no results similar to Theorem 2.1 were established. Such a result is needed to prove consistency. The treatment presented here generalizes their study. Note that for the constrained formulation, it is more difficult to relate $f_c$ at the optimal solution to a probability model, since such a model will have a much more complicated form compared with the unconstrained counterpart.

## 5   Conclusion

In this paper we proposed a family of risk minimization methods for multi-category classification problems, which are natural extensions of binary large margin classification methods. We established admissibility conditions that ensure the consistency of the obtained classifiers in the large sample limit. Two specific forms of risk minimization were proposed and examples were given to study the induced probability models. As an implication of this work, we see that it is possible to obtain consistent (conditional) density estimation using various non-maximum likelihood estimation methods. One advantage of some of the newly proposed methods is that they allow us to model zero density directly. Note that for the maximum-likelihood method, near zero density may cause serious robustness problems at least in theory.

## Footnotes

[1]This approach is often called one-versus-all or ranking in machine learning. Another main approach is to encode a multi-category classification problem into binary classification sub-problems. The consistency of such encoding schemes can be difficult to analyze, and we shall not discuss them.

## References

[1] P.L. Bartlett, M.I. Jordan, and J.D. McAuliffe. Convexity, classification, and risk bounds. Technical Report 638, Statistics Department, University of California, Berkeley, 2003.

[2] Ilya Desyatnikov and Ron Meir. Data-dependent bounds for multi-category classification based on convex losses. In *COLT*, 2003.

[3] J. Friedman, T. Hastie, and R. Tibshirani. Additive logistic regression: A statistical view of boosting. *The Annals of Statistics*, 28(2):337–407, 2000. With discussion.

[4] W. Jiang. Process consistency for adaboost. *The Annals of Statistics*, 32, 2004. with discussion.

[5] Y. Lee, Y. Lin, and G. Wahba. Multicategory support vector machines, theory, and application to the classification of microarray data and satellite radiance data. *Journal of American Statistical Association*, 2002. accepted.

[6] Yi Lin. Support vector machines and the bayes rule in classification. *Data Mining and Knowledge Discovery*, pages 259–275, 2002.

[7] G. Lugosi and N. Vayatis. On the Bayes-risk consistency of regularized boosting methods. *The Annals of Statistics*, 32, 2004. with discussion.

[8] Shie Mannor, Ron Meir, and Tong Zhang. Greedy algorithms for classification - consistency, convergence rates, and adaptivity. *Journal of Machine Learning Research*, 4:713–741, 2003.

[9] Robert E. Schapire and Yoram Singer. Improved boosting algorithms using confidence-rated predictions. *Machine Learning*, 37:297–336, 1999.

[10] Ingo Steinwart. Support vector machines are universally consistent. *J. Complexity*, 18:768–791, 2002.

[11] Tong Zhang. Statistical behavior and consistency of classification methods based on convex risk minimization. *The Annals of Statitics*, 32, 2004. with discussion.
